# Large-Scale Matrix Factorization with Missing Data under Additional Constraints

**Kaushik Mitra** [*†]
Department of Electrical and Computer Engineering and UMIACS
University of Maryland, College Park, MD 20742
kmitra@umiacs.umd.edu

**Sameer Sheorey**[†]
Toyota Technological Institute, Chicago
ssameer@ttic.edu

**Rama Chellappa**
Department of Electrical and Computer Engineering and UMIACS
University of Maryland, College Park, MD 20742
rama@umaics.umd.edu

## Abstract

Matrix factorization in the presence of missing data is at the core of many computer vision problems such as structure from motion (SfM), non-rigid SfM and photometric stereo. We formulate the problem of matrix factorization with missing data as a low-rank semidefinite program (LRSDP) with the advantage that: 1) an efficient quasi-Newton implementation of the LRSDP enables us to solve large-scale factorization problems, and 2) additional constraints such as orthonormality, required in orthographic SfM, can be directly incorporated in the new formulation. Our empirical evaluations suggest that, under the conditions of matrix completion theory, the proposed algorithm finds the optimal solution, and also requires fewer observations compared to the current state-of-the-art algorithms. We further demonstrate the effectiveness of the proposed algorithm in solving the affine SfM problem, non-rigid SfM and photometric stereo problems.

## 1 Introduction

Many computer vision problems such as SfM [26], non-rigid SfM [3] and photometric stereo [11] can be formulated as a matrix factorization problem. In all these problems, the measured data are observations of the elements of an $m \times n$ measurement matrix $M$ of known rank $r$. The objective is to factorize this measurement matrix $M$ into factors $A$ and $B$ of dimensions $m \times r$ and $n \times r$, respectively such that the error $||M - AB^T||$ is minimized. When all the elements of $M$ are known, and assuming that the elements are corrupted by Gaussian noise, the solution to this problem is given by the singular value decomposition (SVD) of $M$. However, in most real applications many of the elements of $M$ will be missing and we need to solve a modified problem given by:

$$\min_{A,B} ||W \odot (M - AB^T)||_F^2 + \lambda_1 ||A||_F^2 + \lambda_2 ||B||_F^2 \qquad (1)$$

where $\odot$ is the Hadamard element-wise product, $W$ is a weight matrix with zeroes at indices corresponding to the missing elements of $M$, and $||A||_F^2$, $||B||_F^2$ are regularization terms which prevent

---

[*]Partially supported by an ARO MURI on oppurtunistic sensing under the grant W911NF-09-1-0383.
[†]Kaushik Mitra and Sameer Sheorey contributed equally to this work.

data overfitting. Matrix factorization with missing data is a difficult non-convex problem with no known globally convergent algorithm. The damped Newton algorithm [4], a variant of Newton's method, is one of the most popular algorithms for solving this problem. However, this algorithm has high computational complexity and memory requirements and so cannot be used for solving large scale problems.

We formulate the matrix factorization with missing data problem as a LRSDP [6], which is essentially a rank constrained semidefinite programming problem (SDP) and was proposed to solve large SDP in an efficient way. The advantages of formulating the matrix factorization problem as a LRSDP problem are the following: 1) it inherits the efficiency of the LRSDP algorithm. The LRSDP algorithm is based on a quasi-Newton method which has lower computational complexity and memory requirements than that of Newton's method, and so is ideally suited for solving large scale problems. 2) Many additional constraints, such as the ortho-normality constraints for the orthographic SfM, can be easily incorporated into the LRSDP-based factorization formulation; this is possible because of the flexible framework of the LRSDP (see section 2).

**Prior Work** Algorithms for matrix factorization in the presence of missing data can be broadly divided into two main categories: initialization algorithms and iterative algorithms. Initialization algorithms [26, 13, 10, 18, 25] generally minimize an algebraic or approximate cost of (1) and are used for providing a good starting point for the iterative algorithms. Iterative algorithms are those algorithms that directly minimize the cost function (1). Alternation algorithms [23, 28, 12, 1, 2, 14], damped Newton algorithm [4] and our approach fall under this category. Alternation algorithms are based on the fact that if one of the factors $A$ or $B$ is known, then there are closed form or numerical solutions for the other factor. Though the alternation-based algorithms minimize the cost in each iteration, they are essentially a coordinate descent approach and suffer from flatlining, requiring an excessive number of iterations before convergence [4]. To solve this problem, damped Newton and hybrid algorithms between damped Newton and alternation were proposed in [4]. Although these algorithms give very good results, they cannot be used for solving large-scale problems because of their high computational complexity and memory requirements. Other algorithms based on Newton's method have been proposed in [7, 21], which also cannot be used for solving large-scale problems.

The matrix factorization with missing data problem is closely related to the matrix completion problem [9]. The goal of matrix completion is to find a low-rank matrix which agrees with the observed entries of the matrix $M$. Recently, many efficient algorithms have been proposed for solving this problem [8, 17, 19, 16, 15, 20]. Some of them [16, 15, 20] are formulated as matrix factorization problems. However, we note that these algorithms, by themselves, can not handle additional constraints. Matrix factorization also arises while solving the collaborative filtering problem. Collaborative filtering is the task of predicting the interests of a user by collecting taste information from many users, for example in a movie recommendation system. In [24], collaborative filtering is formulated as a matrix completion problem and solved using a semidefinite program. Later a fast version, using conjugate gradient, was proposed in [22], but it also cannot handle additional constraints.

## 2 Background: Low-rank semidefinite programming (LRSDP)

LRSDP was proposed in [6] to efficiently solve a large scale SDP [27]. In the following paragraphs, we briefly define the SDP and LRSDP problems, and discuss the efficient algorithm used for solving the LRSDP problem.

SDP is a subfield of convex optimization concerned with the optimization of a linear objective function over the intersection of the cone of positive semidefinite matrices with an affine space. The standard-form SDP is given by:

$$\min \ C \bullet X \text{ subject to } A_i \bullet X = b_i, \quad i = 1, \ldots, k \quad X \succeq 0 \tag{2}$$

where $C$ and $A_i$ are $n \times n$ real symmetric matrices, $b$ is $k$-dimensional vector, and $X$ is an $n \times n$ matrix variable, which is required to be symmetric and positive semidefinite, as indicated by the constraint $X \succeq 0$. The operator $\bullet$ denotes the inner product in the space of $n \times n$ symmetric matrices defined as $A \bullet B = \text{trace}(A^T B) = \sum_{i=1}^{n} \sum_{j=1}^{n} A_{ij} B_{ij}$. The most common algorithms for solving (2) are the interior point methods [27]. However, these are second-order methods, which

need to store and factorize a large (and often dense) matrix and hence are not suitable for solving large scale problems.

In LRSDP a change of variables is introduced as $X = RR^T$, where $R$ is a real, $n \times r$ matrix with $r \leq n$. This has the advantage that it removes the non-linear constraint $X \succeq 0$, which is the most challenging aspect of solving (2). However, this comes with the cost that the problem may no longer be a convex problem. The LRSDP formulation is given by:

$$(\text{N}_r) \quad \min \ C \bullet RR^T \text{ subject to } A_i \bullet RR^T = b_i, \quad i = 1, \ldots, k \tag{3}$$

Note that the LRSDP formulation depends on $r$; when $r = n$, (3) is equivalent to (2). But the intention is to choose $r$ as small as possible so as to reduce the number of variables, while the problem remains equivalent to the original problem (2).

A non-linear optimization technique called the augmented Lagrangian method is used for solving (3). The majority of the iterations in this algorithm involve the minimization of an augmented Lagrangian function with respect to the variable $R$ which is done by a limited memory BFGS method. BFGS, a quasi-Newton method, is much more efficient than Newton's method both in terms of computations and memory requirement. The LRSDP algorithm further optimizes the computations and storage requirements for sparse $C$ and $A_i$ matrices, which is true for problems of our interest. For further details on the algorithm, see [6, 5].

## 3  Matrix factorization using LRSDP (MF-LRSDP)

In this section, we formulate the matrix factorization with missing data as an LRSDP problem. We do this in the following stages: in section 3.1, we look at the noiseless case, that is, where the measurement matrix $M$ is not corrupted with noise, followed by the noisy measurement case in section 3.2, and finally in section 3.3, we look at how additional constraints can be incorporated in the LRSDP formulation.

### 3.1  Noiseless Case

When the observed elements of the $m \times n$ dimensional measurement matrix $M$ are not corrupted with noise, a meaningful cost to minimize would be:

$$\min_{A,B} ||A||_F^2 + ||B||_F^2 \text{ subject to } (AB^T)_{i,j} = M_{i,j} \text{ for } (i,j) \in \Omega \tag{4}$$

where $\Omega$ is the index set of the observed entries of $M$, and $A$, $B$ are the desired factor matrices of dimensions $m \times r$ and $n \times r$ respectively. To formulate this as a LRSDP problem, we introduce a $(m+n) \times r$ dimensional matrix $R = \begin{pmatrix} A \\ B \end{pmatrix}$. Then

$$RR^T = \begin{pmatrix} AA^T & AB^T \\ BA^T & BB^T \end{pmatrix} \tag{5}$$

We observe that the cost function $||A||_F^2 + ||B||_F^2$ can be expressed as $\text{trace}(RR^T)$ and the constraints as $(RR^T)_{i,j+m} = M_{i,j}$. Thus, (4) is equivalent to:

$$\min_R \ \text{trace}(RR^T) \text{ subject to } (RR^T)_{i,j+m} = M_{i,j} \text{ for } (i,j) \in \Omega \tag{6}$$

This is already in the LRSDP form, since we can express the above equation as

$$\min_R \ C \bullet RR^T \text{ subject to } A_l \bullet RR^T = b_l, \quad l = 1, \ldots, |\Omega| \tag{7}$$

where $C$ is an $(m+n) \times (m+n)$ identity matrix, and to simplify the notations we have introduced the index $l$ with $\Omega(l) = (i,j) \quad l = 1, \ldots, |\Omega|$. $A_l$ are sparse matrices with the non-zero entries at indices $(i, j+m)$ and $(j+m, i)$ equal to $1/2$ and $b_l = M_{i,j}$. This completes the formulation of the matrix factorization problem as an LRSDP problem for the noiseless case. Next we look at the noisy case.

## 3.2 Noisy case

When the observed entries of $M$ are corrupted with noise, an appropriate cost function to minimize would be:

$$\min_{A,B} ||W \odot (M - AB^T)||_F^2 + \lambda ||A||_F^2 + \lambda ||B||_F^2 \tag{8}$$

where $\odot$ is the Hadamard element-wise product and $W$ is a weight matrix with zeros corresponding to the missing entries and 1 to the observed entries in $M$. To formulate this as an LRSDP problem, we introduce noise variables $e_l, l = 1, 2, \ldots, |\Omega|$ which are defined as $e_l = (M - (AB^T))_l$. Now, (8) can be expressed as

$$\min_{A,B,e} ||e||_2^2 + \lambda ||A||_F^2 + \lambda ||B||_F^2 \text{ subject to } (M - AB^T)_l = e_l \text{ for } l = 1, 2, \ldots, |\Omega| \tag{9}$$

Next, we aim to formulate this as a LRSDP problem. For this, we construct an augmented noise vector $E = [e^T \quad 1]^T$ and define $R$ to be

$$R = \begin{pmatrix} \begin{pmatrix} A \\ B \end{pmatrix} & 0 \\ 0 & E \end{pmatrix} \tag{10}$$

$R$ is a 'block-diagonal' matrix, where the blocks are of sizes $(m + n) \times r$ and $(|\Omega| + 1) \times 1$ respectively. With this definition, $RR^T$ is a block-diagonal matrix given by

$$RR^T = \begin{pmatrix} \begin{pmatrix} AA^T & AB^T \\ BA^T & BB^T \end{pmatrix} & 0 \\ 0 & EE^T \end{pmatrix} \tag{11}$$

We can now express (8) in the following LRSDP form:

$$\min_R C \bullet RR^T \text{ subject to } A_l \bullet RR^T = b_l, \quad l = 1, \ldots, |\Omega| + 1 \tag{12}$$

with

$$C = \begin{pmatrix} \lambda I_{(m+n) \times (m+n)} & 0 \\ 0 & I_{(|\Omega|+1) \times (|\Omega|+1)} \end{pmatrix} \tag{13}$$

Note that the number of constraints $|\Omega| + 1$ in (12) is one more than the number of observations $|\Omega|$. This is because the last constraint is used to set $E_{|\Omega|+1} = 1$, which is done by choosing $A_{|\Omega|+1}$ to be a sparse matrix with the non-zero entry at index $(|\Omega| + l + m + n, |\Omega| + 1 + m + n)$ equal to 1 and $b_{|\Omega|+1} = 1$. For the remaining values of $l$, the $A_l$ are sparse matrices with the non-zero entries at indices $(i, j+m), (j+m, i), (|\Omega|+1+m+n, l+m+n)$ and $(l+m+n, |\Omega|+1+m+n)$ equal to $1/2$ and $b_l = M_l$. Note that (12) is a block-LRSDP problem ($R$ has a block-diagonal structure), which is a simple extension of the original LRSDP problem [5]. This completes the LRSDP formulation for the noisy case. Next, we look at incorporating additional constraints in this framework.

## 3.3 Enforcing Additional Constraints

Many additional constraints can be easily incorporated in the LRSDP formulation. We illustrate this using the specific example of orthographic SfM [26]. SfM is the problem of reconstructing the scene structure (3-D point positions and camera parameters) from 2-D projections of the points in the cameras. Suppose that $m/2$ cameras are looking at $n$ 3-D points, then under the affine camera model, the 2-D imaged points can be arranged as an $m \times n$ measurement matrix $M$ with columns corresponding to the $n$ 3-D points and rows corresponding to the $m/2$ cameras (2 consecutive rows per camera) [26]. Under this arrangement, $M$ can be factorized as $M = AB^T$, where $A$ is a $m \times 4$ camera matrix and $B$ is a $n \times 4$ structure matrix with the last column of $B$, an all-one vector. Thus, $M$ is a rank 4 matrix with a special structure for the last column of $B$. Further, under the orthographic camera model, $A$ has more structure (constraints): pair of 'rows' that corresponds to the same camera is ortho-normal. To state this constraints precisely, we decompose the $A$ matrix as $A = [P \quad t]$ where $P$ is a $m \times 3$ sub-matrix consisting of the first three columns and $t$ is the last column vector. We can now express the camera ortho-normality constraint through the $PP^T$ matrix, whose diagonal elements should be 1 (normality constraint) and appropriate off-diagonal elements should be 0 (orthogonality constraint). Since, the last column of $B$ is the all one vector, we can write

$B = [X \quad \mathbf{1}]$, where $X$ is a $n \times 3$ matrix. Thus, $AB^T = PX + t\mathbf{1}^T$ and the observation error can be expressed as $e_l = (M - PX)_l - t_i$ for $\Omega(l) = (i,j)$. A meaningful optimization problem to solve here would be to minimize the observation error subject to the ortho-normality constraints:

$$\min_{e,P,X,t} \|e\|_2^2 \quad \text{subject to } e_l = (M - PX)_l - t_i, \quad l = 1, 2, \ldots, |\Omega|$$

$$(PP^T)_{k,k} = 1, \quad k = 1, 2, \ldots, m$$

$$(PP^T)_{k,l} = 0, \text{ if k and l are rows from same camera} \quad (14)$$

To formulate this as an LRSDP problem, we introduce the augmented translation variable $T = [t^T \quad 1]^T$, and propose the following block-diagonal matrix $R$:

$$R = \begin{pmatrix} \begin{pmatrix} P \\ X \end{pmatrix} & 0 & 0 \\ 0 & T & 0 \\ 0 & 0 & E \end{pmatrix} \quad (15)$$

With this definition of $R$, we can express (14) as a LRSDP problem; following steps similar to the previous sections, it is should be straight forward to figure out the appropriate $C$ and $A_l$ matrices required in this LRSDP formulation (3). This completes our illustration on the incorporation of the ortho-normality constraints for the orthographic SfM case. This example should convince the reader that many other application-specific constraints can be directly incorporated into the LRSDP formulation; this is because of the underlying SDP structure of the LRSDP.

# 4   Matrix Completion, Uniqueness and Convergence of MF-LRSDP

In this section, we state the main result of the matrix completion theory and discuss its implications for the matrix factorization problem.

## 4.1   Matrix Completion Theory

Matrix completion theory considers the problem of recovering a low-rank matrix from a few samples of its entries:

$$\min_X \text{ rank}(X) \text{ subject to } X_{i,j} = M_{i,j} \text{ for } (i,j) \in \Omega \quad (16)$$

More specifically, it considers the following questions: 1) when does a partially observed matrix have a unique low-rank solution? 2) How can this matrix be recovered? The answers to these questions were provided in theorem 1.3 of [9] which states that if 1) the matrix $M$, that we want to recover, has row and columns spaces incoherent with the standard basis and 2) we are given enough entries ($\geq O(rd^{6/5}\log d)$, where $d = \max(m,n)$), then there exists a unique low-rank solution to (16). Further, the solution can be obtained by solving a convex relaxation of (16) given by:

$$\min_X \|X\|_* \text{ subject to } X_{i,j} = M_{i,j} \text{ for } (i,j) \in \Omega \quad (17)$$

where $\|X\|_*$ is the nuclear norm of $X$, given by the sum of its singular values.

## 4.2   Relation with Matrix Factorization and its Implications

In matrix completion the objective is to find a minimum rank matrix which agrees with the partial observations (16), whereas in matrix factorization we assume the rank $r$ to be known, as in the problems of SFM and photometric stereo, and we use the rank as a constraint. For example, in our LRSDP formulation, we have imposed this rank constraint by fixing the number of columns of the factors $A$ and $B$ to $r$. However, though the matrix completion and factorization problems are defined differently, they are closely related as revealed by their very similar Lagrangian formulations. This fact has been used in solving the matrix completion problem via matrix factorization with an appropriate rank [16, 15, 20]. We should also note that matrix completion theory helps us answer the question raised in [4]: when is missing data matrix factorization unique (up to a gauge)? And from the discussion in the previous section, it should be clear that the conditions of the matrix completion theory are sufficient for guaranteeing us the required uniqueness. Further, in our experimental evaluations (see next section), we have found that the LRSDP formulation, though a non-convex problem in general, converges to the global minimum solution under these conditions.

# 5 Experimental Evaluation

We evaluate the performance of the proposed LRSDP-based factorization algorithm (MF-LRSDP) on both synthetic and real data and compare it against other algorithms such as alternation [4], damped Newton [4] and OptSpace [15], which is one of state-of-the-art algorithms for matrix completion.

## 5.1 Evaluation with Synthetic Data

The important parameters in the matrix factorization with missing data problem are: the size of the matrix $M$ characterized by $m$ and $n$, rank $r$, fraction of missing data and the variance $\sigma^2$ of the observation noise. We evaluate the factorization algorithms by varying these parameters. We consider two cases: data without noise and data with noise. For synthetic data without noise, we generate $n \times n$ matrices $M$ of rank $r$ by $M = AB^T$, where $A$ and $B$ are $n \times r$ random matrices with each entry being sampled independently from a standard Gaussian distribution $\mathcal{N}(0, 1)$. Each entry is then revealed randomly according to the missing data fraction. For synthetic data with noise, we add independent Gaussian noise $\mathcal{N}(0, \sigma^2)$ to the observed entries generated as above.

**Exact Factorization: a first comparison.** We study the reconstruction rate of different algorithms by varying the fraction of revealed entries per column ($|\Omega|/n$) for noiseless $500 \times 500$ matrices of rank 5. We declare a matrix to be reconstructed if $||M - \hat{M}||_F / ||M||_F \leq 10^{-4}$, where $\hat{M} = \hat{A}\hat{B}$ is the reconstructed matrix and $||.||_F$ denotes the Frobenius norm. Reconstruction rate is defined as the fraction of trials for which the matrix was successfully reconstructed. In all the synthetic data experiments, we performed 10 trials. Figure 1(a) shows the reconstruction rate by MF-LRSDP, alternation and OptSpace. MF-LRSDP gives the best reconstruction results as it needs fewer observations for matrix reconstruction than the other algorithms. It is followed by OptSpace and alternation, respectively. MF-LRSDP also takes the least time, followed by OptSpace and alternation. For similar comparison to other matrix completion algorithms such as ADMiRA [16], SVT [8] and FPCA [17], the interested reader can look at [15], where OptSpace was shown to be consistently better than these algorithms. For the remaining experiments on synthetic data, we mostly compare MF-LRSDP against OptSpace. Note that we have not included the damped Newton algorithm in this comparison because it is very slow for matrices of this size.

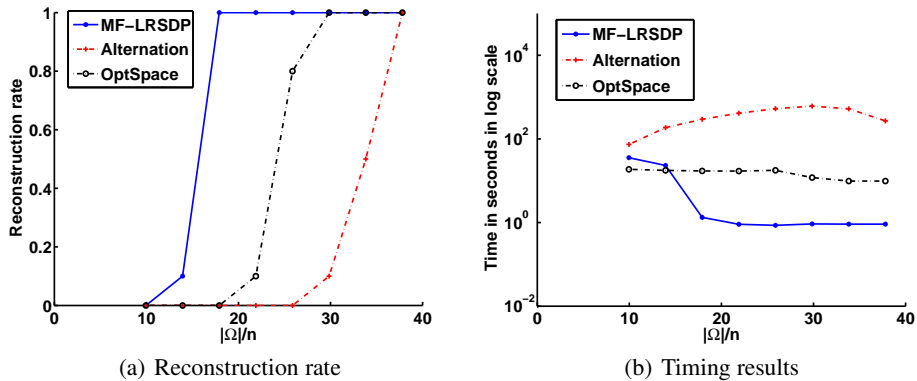

(a) Reconstruction rate        (b) Timing results

Figure 1: (a) Reconstruction rate vs. fraction of revealed entries per column $|\Omega|/n$ for $500 \times 500$ matrices of rank 5 by MF-LRSDP, alternation and OptSpace. The proposed algorithm MF-LRSDP gives the best reconstruction results since it can reconstruct matrices with fewer observed entries. (b) Time taken for reconstruction by different algorithms. MF-LRSDP takes the least time.

**Exact Factorization: vary size.** We study the reconstruction rate vs. fraction of revealed entries per column $|\Omega|/n$ for different sizes $n$ of rank 5 square matrices by MF-LRSDP and OptSpace. Figure 2(a) shows that MF-LRSDP reconstructs matrices from fewer observed entries than OptSpace.

**Exact Factorization: vary rank.** We study the reconstruction rate vs. $|\Omega|/n$ as we vary the rank $r$ of $500 \times 500$ matrices. Figure 2(b) again shows that MF-LRSDP gives better results than OptSpace.

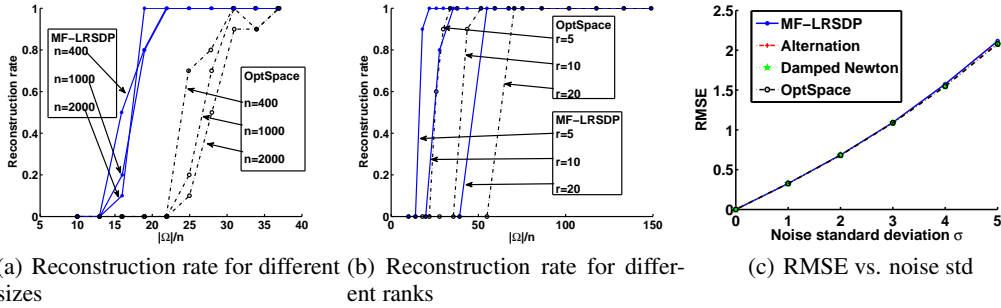

(a) Reconstruction rate for different sizes

(b) Reconstruction rate for different ranks

(c) RMSE vs. noise std

Figure 2: (a) Reconstruction rate vs. fraction of revealed entries per column $|\Omega|/n$ for rank 5 square matrices of different sizes $n$ by MF-LRSDP and OptSpace. MF-LRSDP reconstructs matrices from fewer observed entries than OptSpace. (b) Reconstruction rate vs. $|\Omega|/n$ for $500 \times 500$ matrices of different ranks by MF-LRSDP and OptSpace. Again MF-LRSDP needs fewer observations than OptSpace. (c) RMSE vs. noise standard deviation for rank 5, $200 \times 200$ matrices by MF-LRSDP, OptSpace, alternation and damped Newton. All algorithms perform equally well.

**Noisy Factorization: vary noise standard deviation.** For noisy data, we use the root mean square error RMSE $= 1/\sqrt{mn}||M - \hat{M}||_F$ as a performance measure. We vary the standard deviation $\sigma$ of the additive noise for rank 5, $200 \times 200$ matrices and study the performance by MF-LRSDP, OptSpace, alternation and damped Newton. Figure 2(c) shows that all the algorithms perform equally well.

For timing comparisons, please refer to the supplementary material.

## 5.2   Evaluation with Real Data

We consider three problems: 1) affine SfM 2) non-rigid SfM and 3) photometric stereo.

**Affine SfM.** As discussed in section 3.3, for affine SfM, the $m \times n$ measurement matrix $M$ is a rank 4 matrix with the last column of matrix $B$ an all-one vector. $M$ is generally an incomplete matrix because not all the points are visible in all the cameras. We evaluate the performance of MF-LRSDP on the 'Dinosaur' sequence used in [4, 7], for which $M$ is a $72 \times 319$ matrix with $72\%$ missing entries. We perform 25 trials and at each trial we provide the same random initializations to MF-LRSDP, alternation and damped Newton (OptSpace has its only initialization technique). We use the root mean square error over the observed entries, $||W \odot (M - \hat{M})||_F/\sqrt{|\Omega|}$, as our performance measure. Figure 3 shows the cumulative histogram over the RMS pixel error. MF-LRSDP gives the best performance followed by damped Newton, alternation and OptSpace. We further tested the algorithms on a 'longer Dinosaur', the result of which is provided in the supplementary material.

**Non-rigid SfM.** In non-rigid SfM, non-rigid objects are expressed as a linear combination of $b$ basis shapes. In this case, the $m \times n$ measurement matrix $M$ can be expressed as $M = AB^T$, where $A$ is an $m \times 3b$ matrix and $B$ is an $n \times 3b$ matrix [3]. This makes $M$ a rank $3b$ matrix. We test the performance of the algorithms on the 'Giraffe' sequence [4, 7] for which $M$ is a $240 \times 167$ matrix with $30\%$ missing entries. We choose the rank as 6. Figure 3 shows the cumulative histogram of 25 trials from which we conclude that MF-LRSDP, alternation and damped Newton give good results.

**Photometric Stereo.** Photometric stereo is the problem of estimating the surface normals of an object by imaging that object under different lighting conditions. Suppose we have $n$ images of the object under different lighting conditions with each image consisting of $m$ pixels ($m$ surface normals) and we arrange them as an $m \times n$ measurement matrix $M$. Then under Lambertian assumptions, we can express $M$ as $M = AB^T$, where $A$ is an $m \times 3$ matrix representing the surface normals and reflectance and $B$ is an $n \times 3$ matrix representing the light-source directions and intensities [11]. Thus, $M$ is a rank 3 matrix. Some of the image pixels are likely to be affected by shadows and specularities and those pixels should not be included in the $M$ matrix as they do not obey the Lambertian assumption. This makes $M$, an incomplete matrix. We test the algorithms on the 'Face' sequence [4, 7] for which $M$ is a $2944 \times 20$ matrix with $42\%$ missing entries. The cumulative histogram in figure 3 shows that MF-LRSDP and damped Newton gives the best results followed by alternation and OptSpace.

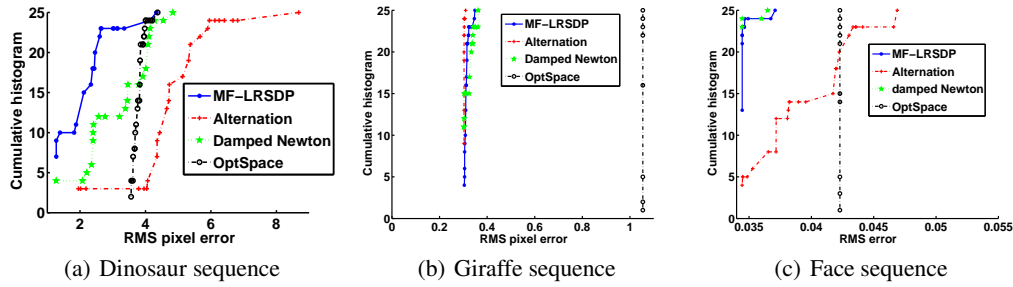

| (a) Dinosaur sequence | (b) Giraffe sequence | (c) Face sequence |

Figure 3: Cumulative histogram (of 25 trials) for the Dinosaur, Giraffe and the Face sequence. For all of them, MF-LRSDP consistently gives good results.

**Additional constraints: Orthographic SfM.** Orthographic SfM is a special case of affine SfM, where the camera matrix $A$ satisfies the additional constraint of ortho-normality, see section 3.3. We show here that incorporating these constraints leads to a better solution. Figure 4 shows the input point tracks, reconstructed point tracks without the constraints and reconstructed point tracks with the constraints for the Dinosaur turntable sequence. Without the constraints many tracks fail to be circular, whereas with the constraints all of them are circular (the dinosaur sequence is a turntable sequence and the tracks are supposed to be circular). Thus, incorporating all the constraints of a problem leads to better solution and MR-LRSDP provides a very flexible framework for doing so.

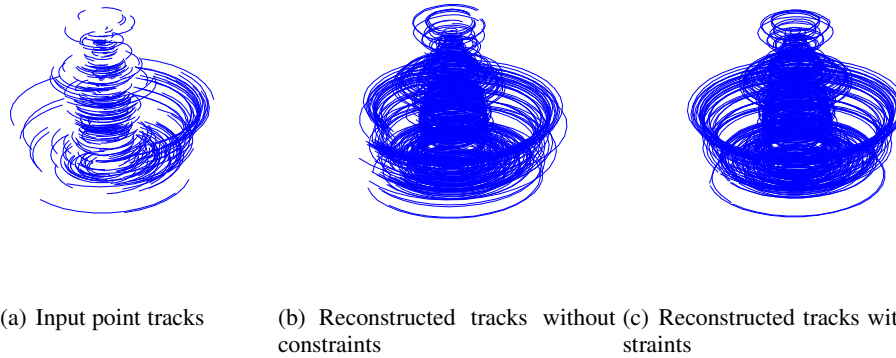

| (a) Input point tracks | (b) Reconstructed tracks without constraints | (c) Reconstructed tracks with constraints |

Figure 4: (a) Input (incomplete) point tracks of the Dinosaur turntable sequence, (b) reconstructed tracks without orthonormality constraints and (c) reconstructed tracks with orthonormality contraints. Without the constraints many tracks fail to be circular, whereas with the constraints all of them are circular (the dinosaur sequence is a turntable sequence and the tracks are supposed to be circular).

## 6 Conclusion and Discussion

We have formulated the matrix factorization with missing data problem as a low-rank semidefinite programming problem MF-LRSDP. MF-LRSDP is an efficient algorithm that can be used for solving large-scale factorization problems. It is also flexible for handling many additional constraints such as the ortho-normality constraints of orthographic SfM. Our empirical evaluations on synthetic data show that it needs fewer observations for matrix factorization as compared to other algorithms and it gives very good results on the real problems of SfM, non-rigid SfM and photometric stereo. We note that though MF-LRSDP is a non-convex problem, it finds the global minimum under the conditions of matrix completion theory. As a future work, it would be interesting to find a theoretical justification for this. Also, it would be interesting to find out how MF-LRSDP performs on collaborative filtering problems.

## References

[1] H. Aanæs, R. Fisker, K. Åström, and J. M. Carstensen. Robust factorization. *IEEE TPAMI*, 2002.

[2] S. Brandt. Closed-form solutions for affine reconstruction under missing data. In *Stat. Methods for Video Proc. (ECCV 02 Workshop)*, 2002.

[3] C. Bregler, A. Hertzmann, and H. Biermann. Recovering non-rigid 3d shape from image streams. In *CVPR*, 2000.

[4] A. M. Buchanan and A. W. Fitzgibbon. Damped newton algorithms for matrix factorization with missing data. In *CVPR*, 2005.

[5] S. Burer and C. Choi. Computational enhancements in low-rank semidefinite programming. *Optimization Methods and Software*, 2006.

[6] S. Burer and R.D.C. Monteiro. A nonlinear programming algorithm for solving semidefinite programs via low-rank factorization. *Mathematical Programming (series B*, 2001.

[7] Pei C. Optimization algorithms on subspaces: Revisiting missing data problem in low-rank matrix. *IJCV*, 2008.

[8] J. Cai, E. J. Candès, and Z. Shen. A singular value thresholding algorithm for matrix completion. *SIAM Journal on Optimization*, 2010.

[9] E. J. Candès and B. Recht. Exact matrix completion via convex optimization. *Foundations on Computational Mathematics*, 2009.

[10] N. Guilbert, A.E. Bartoli, and A. Heyden. Affine approximation for direct batch recovery of euclidian structure and motion from sparse data. *IJCV*, 2006.

[11] H. Hayakawa. Photometric stereo under a light source with arbitrary motion. *JOSA*, 1994.

[12] D. Q. Huynh, R. Hartley, and A. Heyden. Outlier correction in image sequences for the affine camera. In *ICCV*, 2003.

[13] D. W. Jacobs. Linear fitting with missing data for structure-from-motion. *CVIU*, 2001.

[14] Q. Ke and T. Kanade. Robust l1 norm factorization in the presence of outliers and missing data by alternative convex programming. In *CVPR*, 2005.

[15] R. H. Keshavan and S. Oh. A gradient descent algorithm on the grassman manifold for matrix completion. *CoRR, abs/0910.5260*, 2009.

[16] K. Lee and Y. Bresler. Admira: Atomic decomposition for minimum rank approximation. *CoRR, abs/0905.0044*, 2009.

[17] S. Ma, D. Goldfarb, and L. Chen. Fixed point and bregman iterative methods for matrix rank minimization. *Mathematical Programming*, 2009.

[18] D. Martinec and T. Pajdla. 3d reconstruction by fitting low-rank matrices with missing data. In *CVPR*, 2005.

[19] R. Mazumder, T. Hastie, and R. Tibshirani. Spectral regularization algorithms for learning large incomplete matrices. *http://www-stat.stanford.edu/ hastie/Papers/SVD_JMLR.pdf*, 2009.

[20] R. Meka, P. Jain, and I. S. Dhillon. Guaranteed rank minimization via singular value projection. *CoRR*, abs/0909.5457, 2009.

[21] T. Okatani and K. Deguchi. On the wiberg algorithm for matrix factorization in the presence of missing components. *IJCV*, 2007.

[22] J. D. M. Rennie and N. Srebro. Fast maximum margin matrix factorization for collaborative prediction. In *ICML*, 2005.

[23] H. Shum, K. Ikeuchi, and R. Reddy. Principal component analysis with missing data and its application to polyhedral object modeling. *IEEE TPAMI*, 1995.

[24] N. Srebro, J. D. M. Rennie, and T. Jaakkola. Maximum-margin matrix factorization. In *NIPS*, 2004.

[25] J. P. Tardif, A. Bartoli, M. Trudeau, N. Guilbert, and S. Roy. Algorithms for batch matrix factorization with application to structure-from-motion. In *CVPR*, 2007.

[26] C. Tomasi and T. Kanade. Shape and motion from image streams under orthography: a factorization method. *IJCV*, 1992.

[27] L. Vandenberghe and S. Boyd. Semidefinite programming. *SIAM Rev.*, 1996.

[28] R. Vidal and R. Hartley. Motion segmentation with missing data using powerfactorization and gpca. In *In CVPR*, 2004.

